# Directed Graph Embedding: an Algorithm based on Continuous Limits of Laplacian-type Operators

**Dominique C. Perrault-Joncas**
Department of Statistics
University of Washington
Seattle, WA 98195
dcpj@stat.washington.edu

**Marina Meilă**
Department of Statistics
University of Washington
Seattle, WA 98195
mmp@stat.washington.edu

## Abstract

This paper considers the problem of embedding directed graphs in Euclidean space while retaining directional information. We model the observed graph as a sample from a manifold endowed with a vector field, and we design an algorithm that separates and recovers the features of this process: the geometry of the manifold, the data density and the vector field. The algorithm is motivated by our analysis of Laplacian-type operators and their continuous limit as generators of diffusions on a manifold. We illustrate the recovery algorithm on both artificially constructed and real data.

## 1   Motivation

Recent advances in graph embedding and visualization have focused on undirected graphs, for which the graph Laplacian properties make the analysis particularly elegant [1, 2]. However, there is an important number of graph data, such as social networks, alignment scores between biological sequences, and citation data, which are naturally asymmetric. A commonly used approach for this type of data is to disregard the asymmetry by studying the spectral properties of $W + W^T$ or $W^T W$, where $W$ is the affinity matrix of the graph.

Some approaches have been offered to preserve the asymmetry information contained in data: [3], [4], [5] or to define directed Laplacian operators [6]. Although quite successful, these works adopt a purely graph-theoretical point of view. Thus, they are not concerned with the generative process that produces the graph, nor with the interpretability and statistical properties of their algorithms.

In contrast, we view the nodes of a directed graph as a finite sample from a manifold in Euclidean space, and the edges as macroscopic observations of a diffusion kernel between neighboring points on the manifold. We explore how this diffusion kernel determines the overall connectivity and asymmetry of the resulting graph and demonstrate how Laplacian-type operators of this graph can offer insights into the underlying generative process.

Based on the analysis of the Laplacian-type operators, we derive an algorithm that, in the limit of infinite sample and vanishing bandwidth, recovers the key features of the sampling process: manifold geometry, sampling distribution, and local directionality, up to their intrinsic indeterminacies.

## 2   Model

The first premise here is that we observe a directed graph $G$, with $n$ nodes, having weights $W = [W_{ij}]$ for the edge from node $i$ to node $j$. In following with common Laplacian-based embedding approaches, we assume that $G$ is a geometric random graph constructed from $n$ points sampled according to distribution $p = e^{-U}$ on an unobserved compact smooth manifold $\mathcal{M} \subseteq \mathbb{R}^l$ of known intrinsic dimension $d \leq l$. The edge weight $W_{ij}$ is then determined by a directed similarity kernel $k_\epsilon(x_i, x_j)$ with bandwidth $\epsilon$. The directional component of $k_\epsilon(x_i, x_j)$ will be taken to be derived

from a vector field $\mathbf{r}$ on $\mathcal{M}$, which assigns a preferred direction between weights $W_{ij}$ and $W_{ji}$. The choice of a vector field $\mathbf{r}$ to characterize the directional component of $G$ might seem restrictive at first. In the asymptotic limit of $\epsilon \to 0$ and $n \to \infty$ however, kernels are characterized by their diffusion, drift, and source components [7]. As such, $\mathbf{r}$ is sufficient to characterize any directionality associated with a drift component and as it turns out, the component of $\mathbf{r}$ normal $\mathcal{M}$ in $\mathbb{R}^l$ can also be use to characterize any source component. As for the diffusion component, it is not possible to uniquely identify it from $G$ alone [8]. Some absolute knownledge of $\mathcal{M}$ is needed to say anything about it. Hence, without loss of generality, we will construct $k_\epsilon(x_i, x_j)$ so that the diffusion component ends being isotropic and constant, i.e. equal to Laplace-Beltrami operator $\Delta$ on $\mathcal{M}$.

The schematic of this generative process is shown in the top left of Figure 1 below.

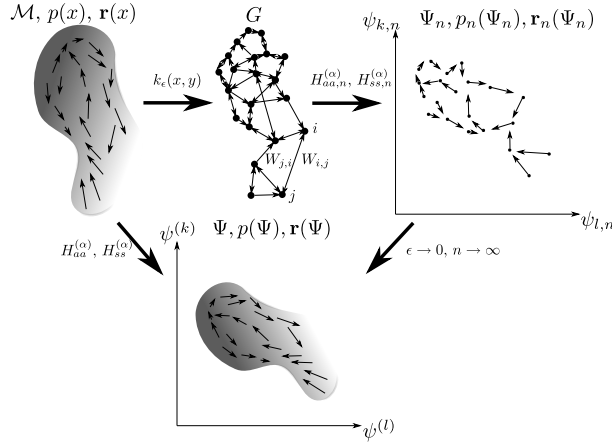

From left to right: the graph generative process mapping the sample on $\mathcal{M}$ to geometric random graph $G$ via the kernel $k_\epsilon(x, y)$, then the subsequent embedding $\Psi_n$ of $G$ by operators $H_{aa,n}^{(\alpha)}$, $H_{ss,n}^{(\alpha)}$ (defined in section 3.1). As these operators converge to their respective limits, $H_{aa}^{(\alpha)}$ and $H_{ss}^{(\alpha)}$, so will $\Psi_n \to \Psi$, $p_n \to p$, and $\mathbf{r}_n \to \mathbf{r}$.

We design an algorithm that, given $G$, produces the top right embedding ($\Psi_n$, $p_n$, and $\mathbf{r}_n$).

Figure 1: Schematic of our framework.

The question is then as follows: can the generative process' geometry $\mathcal{M}$, distribution $p = e^{-U}$, and directionality $\mathbf{r}$, be recovered from $G$? In other words, is there an embedding of $G$ in $\mathbb{R}^m$, $m \geq d$ that approximates all three components of the process and that is also consistent as sample size increases and the bandwidth vanishes? In the case of undirected graphs, the theory of Laplacian eigenmaps [1] and Diffusion maps [9] answers this question in the affirmative, in that the geometry of $\mathcal{M}$ and $p = e^{-U}$ can be inferred using spectral graph theory. The aim here is to build on the undirected problem and recover all three components of the generative process from a directed graph $G$.

The spectral approach to undirected graph embedding relies on the fact that eigenfunctions of the Laplace-Beltrami operator are known to preserve the local geometry of $\mathcal{M}$ [1]. With a consistent empirical Laplace-Beltrami operator based on $G$, its eigenvectors also recover the geometry of $\mathcal{M}$ and converge to the corresponding eigenfunctions on $\mathcal{M}$. For a directed graph $G$, an additional operator is needed to recover the local directional component $\mathbf{r}$, but the principle remains the same. The schematic for this is shown in Figure 1 where two operators - $H_{ss,n}^{(\alpha)}$, introduced in [9] for undirected embeddings, and $H_{aa,n}^{(\alpha)}$, a new operator defined in section 3.1 - are used to obtain the embedding $\Psi_n$, distribution $p_n$, and vector field $\mathbf{r}_n$. As $H_{aa,n}^{(\alpha)}$ and $H_{ss,n}^{(\alpha)}$ converge to $H_{aa}^{(\alpha)}$ and $H_{ss}^{(\alpha)}$, $\Psi_n$, $p_n$, and $\mathbf{r}_n$ also converge to $\Psi$, $p$, and $\mathbf{r}$, where $\Psi$ is the local geometry preserving the embedding of $\mathcal{M}$ into $\mathbb{R}^m$.

The algorithm we propose in Section 4 will calculate the matrices corresponding to $H_{\cdot,n}^{(\alpha)}$ from the graph $G$, and with their eigenvectors, will find estimates for the node coordinates $\Psi$, the directional component $\mathbf{r}$, and the sampling distribution $p$. In the next section we briefly describe the mathematical models of the diffusion processes that our model relies on.

## 2.1 Problem Setting

The similarity kernel $k_\epsilon(x, y)$ can be used to define transport operators on $\mathcal{M}$. The natural transport operator is defined by normalizing $k_\epsilon(x, y)$ as

$$T_\epsilon[f](x) = \int_{\mathcal{M}} \frac{k_\epsilon(x, y)}{p_\epsilon(x)} f(y) p(y) dy, \text{ where } p_\epsilon(x) = \int_{\mathcal{M}} k_\epsilon(x, y) p(y) dy. \tag{1}$$

$T_\epsilon[f](x)$ represents the diffusion of a distribution $f(y)$ by the transition density $k_\epsilon(x, y)p(y) / \int k_\epsilon(x, y')p(y')dy'$. The eigenfunctions of this infinitesimal operator are the continuous limit of the eigenvectors of the transition probability matrix $P = D^{-1}W$ given by normalizing the affinity matrix $W$ of $G$ by $D = \text{diag}(W\mathbf{1})$ [10]. Meanwhile, the infinitesimal transition

$$\frac{\partial f}{\partial t} = \lim_{\epsilon \to 0} \frac{(T_\epsilon - I)f}{\epsilon} \tag{2}$$

defines the backward equation for this diffusion process over $\mathcal{M}$ based on kernel $k_\epsilon$. Obtaining the explicit expression for transport operators like (2) is then the main technical challenge.

## 2.2 Choice of Kernel

In order for $T_\epsilon[f]$ to have the correct asymptotic form, some hypotheses about the similarity kernel $k_\epsilon(x, y)$ are required. The hypotheses are best presented by considering the decomposition of $k_\epsilon(x, y)$ into symmetric $h_\epsilon(x, y) = h_\epsilon(y, x)$ and anti-symmetric $a_\epsilon(x, y) = -a_\epsilon(y, x)$ components:

$$k_\epsilon(x, y) = h_\epsilon(x, y) + a_\epsilon(x, y). \tag{3}$$

The symmetric component $h_\epsilon(x, y)$ is assumed to satisfy the following properties: 1. $h_\epsilon(||y - x||^2) = \frac{h(||y-x||^2/\epsilon)}{\epsilon^{d/2}}$, and 2. $h \geq 0$ and $h$ is exponentially decreasing as $||y - x|| \to \infty$. This form of symmetric kernel was used in [9] to analyze the diffusion map. For the asymmetric part of the similarity kernel, we assume the form

$$a_\epsilon(x, y) = \frac{\mathbf{r}(x, y)}{2} \cdot (y - x) \frac{h(||y - x||^2/\epsilon)}{\epsilon^{d/2}}, \tag{4}$$

with $\mathbf{r}(x, y) = \mathbf{r}(y, x)$ so that $a_\epsilon(x, y) = -a_\epsilon(y, x)$. Here $\mathbf{r}(x, y)$ is a smooth vector field on the manifold that gives an orientation to the asymmetry of the kernel $k_\epsilon(x, y)$. It is worth noting that the dependence of $\mathbf{r}(x, y)$ on both $x$ and $y$ implies that $\mathbf{r} : \mathcal{M} \times \mathcal{M} \to \mathbb{R}^l$ with $\mathbb{R}^l$ the ambient space of $\mathcal{M}$; however in the asymptotic limit, the dependence in $y$ is only important "locally" ($x = y$), and as such it is appropriate to think of $\mathbf{r}(x, x)$ being a vector field on $\mathcal{M}$. As a side note, it is worth pointing out that even though the form of $a_\epsilon(x, y)$ might seem restrictive at first, it is sufficiently rich to describe any vector field . This can be seen by taking $\mathbf{r}(x, y) = (\mathbf{w}(x) + \mathbf{w}(y))/2$ so that at $x = y$ the resulting vector field is given by $\mathbf{r}(x, x) = \mathbf{w}(x)$ for an arbitrary vector field $\mathbf{w}(x)$.

## 3 Continuous Limit of Laplacian Type Operators

We are now ready to state the main asymptotic result.

**Proposition 3.1** *Let $\mathcal{M}$ be a compact, closed, smooth manifold of dimension $d$ and $k_\epsilon(x, y)$ an asymmetric similarity kernel satisfying the conditions of section 2.2, then for any function $f \in C^2(\mathcal{M})$, the integral operator based on $k_\epsilon$ has the asymptotic expansion*

$$\int_{\mathcal{M}} k_\epsilon(x, y) f(y) dy = m_0 f(x) + \epsilon g(f(x), x) + o(\epsilon), \tag{5}$$

*where*

$$g(f(x), x) = \frac{m_2}{2} \left( \omega(x) f(x) + \Delta f(x) + \mathbf{r} \cdot \nabla f(x) + f(x) \nabla \cdot \mathbf{r} + c(x) f(x) \right) \tag{6}$$

*and $m_0 = \int_{\mathbb{R}^d} h(||u||^2) du$, $m_2 = \int_{\mathbb{R}^d} u_i^2 h(||u||^2) du$.*

The proof can be found in [8] along with the definition of $\omega(x)$ and $c(x)$ in (6). For now, it suffices to say that $\omega(x)$ corresponds to an interaction between the symmetric kernel $h_\epsilon$ and the curvature of $\mathcal{M}$ and was first derived in [9]. Meanwhile, $c(x)$ is a new term that originates from the interaction between $h_\epsilon$ and the component of $\mathbf{r}$ that is normal to $\mathcal{M}$ in the ambient space $\mathbb{R}^l$. Proposition 3.1 foreshadows a general fact about spectral embedding algorithms: in most cases, Laplacian operators confound the effects of spatial proximity, sampling density and directional flow due to the presence of the various terms above.

## 3.1 Anisotropic Limit Operators

Proposition 3.1 above can be used to derive the limits of a variety of Laplacian type operators associated with spectral embedding algorithms like [5, 6, 3]. Although we will focus primarily on a few operators that give the most insight into the generative process and enable us to recover the model defined in Figure 1, we first present four distinct families of operators for completeness.

These operator families are inspired by the anisotropic family of operators that [9] introduced for undirected graphs, which make use of anisotropic kernels of the form:

$$k_\epsilon^{(\alpha)}(x, y) = \frac{k_\epsilon(x, y)}{p_\epsilon^\alpha(x) p_\epsilon^\alpha(y)}, \tag{7}$$

with $\alpha \in [0, 1]$ where $\alpha = 0$ is the isotropic limit. To normalize the anisotropic kernels, we need to redefine the outdegrees distribution of $k_\epsilon^{(\alpha)}$ as $p_\epsilon^{(\alpha)}(x) = \int_\mathcal{M} k_\epsilon^{(\alpha)}(x, y) p(y) dy$. From (7), four families of diffusion processes of the form $f_t = H^{(\alpha)}[f](x)$ can be derived depending on which kernel is normalized and which outdegree distribution is used for the normalization. Specifically, we define transport operators by normalizing the asymmetric $k_\epsilon^{(\alpha)}$ or symmetric $h_\epsilon^{(\alpha)}$ kernels with the asymmetric $p_\epsilon$ or symmetric $q_\epsilon = \int_\mathcal{M} h_\epsilon(x, y) p(y) dy$ outdegree distribution[1]. To keep track of all options, we introduce the following notation: the operators will be indexed by the type of kernel and outdegree distribution they correspond to (symmetric or asymmetric), with the first index identifying the kernel and the second index identifying the outdegree distribution. For example, the family of anisotropic limit operators introduced by [9] is defined by normalizing the symmetric kernel by the symmetric outdegree distribution, hence they will be denoted as $H_{ss}^{(\alpha)}$, with the superscript corresponding to the anisotropic power $\alpha$.

**Proposition 3.2** *With the above notation,*

$$
\begin{align}
H_{aa}^{(\alpha)}[f] &= \Delta f - 2(1-\alpha)\nabla U \cdot \nabla f + \mathbf{r} \cdot \nabla f \tag{8}\\
H_{as}^{(\alpha)}[f] &= \Delta f - 2(1-\alpha)\nabla U \cdot \nabla f - cf + (\alpha-1)(\mathbf{r} \cdot \nabla U)f - (\nabla \cdot \mathbf{r})f + \mathbf{r} \cdot \nabla f \tag{9}\\
H_{sa}^{(\alpha)}[f] &= \Delta f - 2(1-\alpha)\nabla U \cdot \nabla f + (c + \nabla \cdot r + (\alpha-1)\mathbf{r} \cdot \nabla U)f \tag{10}\\
H_{ss}^{(\alpha)}[f] &= \Delta f - 2(1-\alpha)\nabla U \cdot \nabla f. \tag{11}
\end{align}
$$

The proof of this proposition, which can be found in [8], follows from repeated application of Proposition 3.1 to $p(y)$ or $q(y)$ and then to $k^\alpha(x, y)$ or $h^\alpha(x, y)$, as well as the fact that $\frac{1}{p_\epsilon^\alpha} = \frac{1}{p^{-\alpha}}[1 - \alpha\epsilon(\omega + \frac{\Delta p}{p} + 2r \cdot \frac{\nabla p}{p} + 2\nabla \cdot r + c)] + o(\epsilon)$.

Thus, if we use the asymmetric $k_\epsilon$ and $p_\epsilon$, we get $H_{aa}^{(\alpha)}$, defined by the advected diffusion equation (8). In general, $H_{aa}^{(\alpha)}$ is not hermitian, so it commonly has complex eigenvectors. This makes embedding directed graphs with this operator problematic. Nevertheless, $H_{aa}^{(1)}$ will play an important role in extracting the directionality of the sampling process.

If we use the symmetric kernel $h_\epsilon$ but the asymmetric outdegree distribution $p_\epsilon$, we get the family of operators $H_{sa}^{(\alpha)}$, of which the WCut of [3] is a special case ($\alpha = 0$). If we reverse the above, i.e. use $k_\epsilon$ and $q_\epsilon$, we obtain $H_{as}^{(\alpha)}$. This turns out to be merely a combination of $H_{aa}^{(\alpha)}$ and $H_{sa}^{(\alpha)}$.

**Algorithm 1** Directed Embedding

---

**Input:** Affinity matrix $W_{i,j}$ and embedding dimension $m$, $(m \geq d)$
1. $S \leftarrow (W + W^T)/2$ (*Steps 1–6 estimate the coordinates as in [11]*)
2. $q_i \leftarrow \sum_{j=1}^{n} S_{i,j}, \ Q = diag(q)$
3. $V \leftarrow Q^{-1}SQ^{-1}$
4. $q_i^{(1)} \leftarrow \sum_{j=1}^{n} V_{i,j}, \ Q^{(1)} = diag(q^{(1)})$
5. $H_{ss,n}^{(1)} \leftarrow Q^{(1)^{-1}}V$
6. Compute the $\Psi$ the $n \times (m+1)$ matrix with orthonormal columns containing the $m+1$ largest right eigenvector (by eigenvalue) of $H_{ss,n}^{(1)}$ as well as the $\Lambda$ the $(m+1) \times (m+1)$ diagonal matrix of eigenvalues. Eigenvectors 2 to $m+1$ from $\Psi$ are the $m$ coordinates of the embedding.
7. Compute $\pi$ the left eigenvector of $H_{ss,n}^{(1)}$ with eigenvalue 1. (*Steps 7–8 estimate the density*)
8. $\pi \leftarrow \pi / \sum_{i=1}^{n} \pi_i$ is the density distribution over the embedding.
9. $p_i \leftarrow \sum_{j=1}^{n} W_{i,j}, \ P = diag(p)$ (*Steps 9–13 estimate the vector field* $\mathbf{r}$)
10. $T \leftarrow P^{-1}WP^{-1}$
11. $p_i^{(1)} \leftarrow \sum_{j=1}^{n} T_{i,j}, \ P^{(1)} = diag(p^{(1)})$
12. $H_{aa,n}^{(1)} \leftarrow P^{(1)^{-1}}T$
13. $R \leftarrow (H_{aa,n}^{(1)} - H_{ss,n}^{(1)})\Psi/2$. Columns 2 to $m+1$ of $R$ are the vector field components in the direction of the corresponding coordinates of the embedding.

---

Finally, if we only consider the symmetric kernel $h_\epsilon$ and degree distribution $q_\epsilon$, we recover $H_{ss}^{(\alpha)}$, the anisotropic kernels of [9] for symmetric graphs. This operator for $\alpha = 1$ is shown to separate the manifold from the probability distribution [11] and will be used as part of our recovery algorithm.

## 4 Isolating the Vector Field r

Our aim is to esimate the manifold $\mathcal{M}$, the density distribution $p = e^{-U}$, and the vector field $\mathbf{r}$. The first two components of the data can be recovered from $H_{ss}^{(1)}$ as shown in [11] and summarized in Algorithm 1.

At this juncture, one feature of generative process is missing: the vector field $\mathbf{r}$. The natural approach for recovering $\mathbf{r}$ is to isolate the linear operator $\mathbf{r} \cdot \nabla$ from $H_{aa}^{(\alpha)}$ by substracting $H_{ss}^{(\alpha)}$:

$$H_{aa}^{(\alpha)} - H_{ss}^{(\alpha)} = \mathbf{r} \cdot \nabla. \tag{12}$$

The advantage of recovering $\mathbf{r}$ in operator form as in (12) is that $\mathbf{r} \cdot \nabla$ is *coordinate free*. In other words, as long as the chosen embedding of $\mathcal{M}$ is diffeomorphic to $\mathcal{M}^2$, (12) can be used to express the component of $\mathbf{r}$ that lies in the tangent space $T\mathcal{M}$, which we denote by $\mathbf{r}_{||}$.

Specifically, let $\Psi$ be a diffeomorphic embedding of $\mathcal{M}$; the component of $\mathbf{r}$ along coordinate $\psi_k$ is then given by $\mathbf{r} \cdot \nabla \psi_k = r_k$, and so, in general,

$$\mathbf{r}_{||} = \mathbf{r} \cdot \nabla \Psi. \tag{13}$$

The subtle point that only $\mathbf{r}_{||}$ is recovered from (13) follows from the fact that the operator $\mathbf{r} \cdot \nabla$ is only defined along $\mathcal{M}$ and hence any directional derivative is necessarily along $T\mathcal{M}$.

Equation (13) and the previous observations are the basis for Algorithm 1, which recovers the three important features of the generative process for an asymmetric graph with affinity matrix $W$.

A similar approach can be employed to recover $c + \nabla \cdot \mathbf{r}$, or simply $\nabla \cdot \mathbf{r}$ if $\mathbf{r}$ has no component perpendicular to the tangent space $T\mathcal{M}$ (meaning that $c \equiv 0$). Recovering $c + \nabla \cdot \mathbf{r}$ is achieved by taking advantage of the fact that

$$(H_{sa}^{(1)} - H_{ss}^{(1)}) = (c + \nabla \cdot \mathbf{r}), \tag{14}$$

which is a diagonal operator. Taking into account that for finite $n$ $(H_{sa,n}^{(1)} - H_{ss,n}^{(1)})$ is not perfectly diagonal, using $\psi_n \equiv 1_n$ (vector of ones), i.e. $(H_{sa,n}^{(1)} - H_{ss,n}^{(1)})[1_n] = (c_n + \nabla \cdot \mathbf{r}_n)$, has been found empirically to be more stable than simply extracting the diagonal of $(H_{sa,n}^{(1)} - H_{ss,n}^{(1)})$.

## 5    Experiments

**Artificial Data** For illustrative purposes, we begin by applying our method to an artificial example. We use the planet Earth as a manifold with a topographic density distribution, where sampling probability is proportional to elevation. We also consider two vector fields: the first is parallel to the line of constant latitude and purely tangential to the sphere, while the second is parallel to the line of constant longitude with a component of the vector field perpendicular to the manifold. The true model with constant latitude vector field is shown in Figure 2, along with the estimated density and vector field projected on the true manifold (sphere).

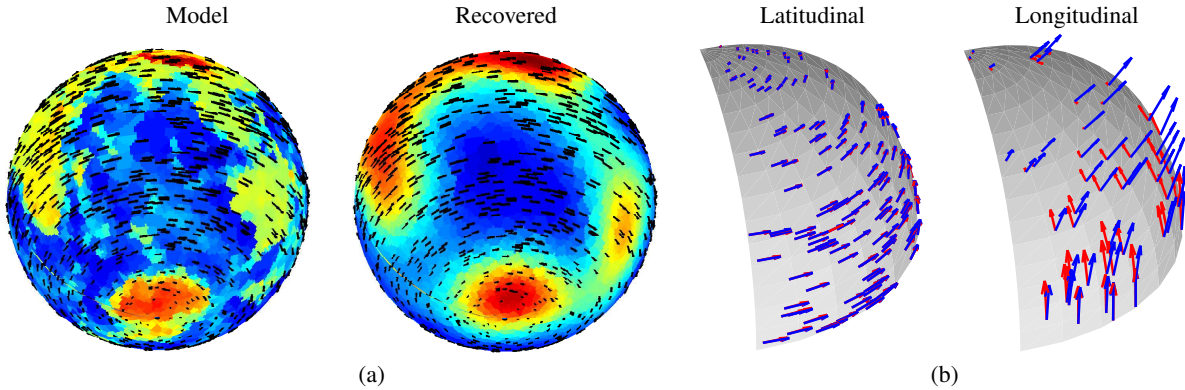

| Model | Recovered | Latitudinal | Longitudinal |

(a)                                                                              (b)

Figure 2:    (a): Sphere with latitudinal vector field, i.e East-West asymmetry, with $W_{ew} > W_{we}$ if node $w$ lies to the West of node $e$. The graph nodes are sampled non-uniformly, with the topographic map of the world as sampling density. We sample $n = 5000$ nodes, and observe only the resulting $W$ matrix, but not the node locations. From $W$, our algorithm estimates the sample locations (geometry), the vector field (black arrows) generating the observed asymmetries, and the sampling distribution at each data point (colormap). (b) Vector fields on a spherical region (blue), and their estimates (red): latitudinal vector field tangent to the manifold (left) and longitudinal vector field with component perpendicular to manifold tangent plane (right).

Both the estimated density and vector field agree with the true model, demonstrating that for artificial data, the recovery algorithm 1 performs quite well. We note that the estimated density does not recover all the details of the original density, even for large sample size (here $n = 5000$ with $\epsilon = 0.07$). Meanwhile, the estimated vector field performs quite well even when the sampling is reduced to $n = 500$ with $\epsilon = 0.1$. This can be seen in Figure 2, b, where the true and estimated vector fields are superimposed. Figure 2 also demonstrates how $\mathbf{r} \cdot \nabla$ only recovers the tangential component of $\mathbf{r}$. The estimated geometry is not shown on any of these figures, since the success of the diffusion map in recovering the geometry for such a simple manifold is already well established [2, 9].

**Real Data** The National Longitudinal Survey of Youth (NLSY) 1979 Cohort is a representative sample of young men and women in the United States who were followed from 1979 to 2000 [12, 13]. The aim here is to use this survey to obtain a representation of the job market as a diffusion process over a manifold.

The data set consists of a sample of 7,816 individual career sequences of length 64, listing the jobs a particular individual held every quarter between the ages of 20 and 36. Each *token* in the sequence identifies a job. Each job corresponds to an *industry* $\times$ *occupation* pair. There are 25 unique industry and 20 unique occupation indices. Out of the 500 possible pairings, approximately 450 occur in the data, with only 213 occurring with sufficient frequency to be included here. Thus, our graph $G$ has 213 nodes - the jobs - and our observations consist of 7,816 walks between the graph nodes.

We convert these walks to a directed graph with *affinity matrix* $W$. Specifically, $W_{ij}$ represents the number of times a transition from job $i$ to job $j$ was observed (Note that this matrix is *asymmetric*,

i.e $W_{ij} \neq W_{ji}$). Normalizing each row $i$ of $W$ by its outdegree $d_i$ gives $P = \mathrm{diag}(d_i)^{-1}W$, the non-parametric maximum likelihood estimator for the Markov chain over $G$ for the progression of career sequences. This Markov chain has as limit operator $H_{aa}^{(0)}$, as the granularity of the job market increases along with the number of observations. Thus, in trying to recover the geometry, distribution and vector field, we are actually interested in estimating the full advective effect of the diffusion process generated by $H_{aa}^{(0)}$; that is, we want to estimate $\mathbf{r} \cdot \nabla - 2\nabla U \cdot \nabla$ where we can use $-2\nabla U \cdot \nabla = H_{ss}^{(0)} - H_{ss}^{(1)}$ to complement Algorithm 1.

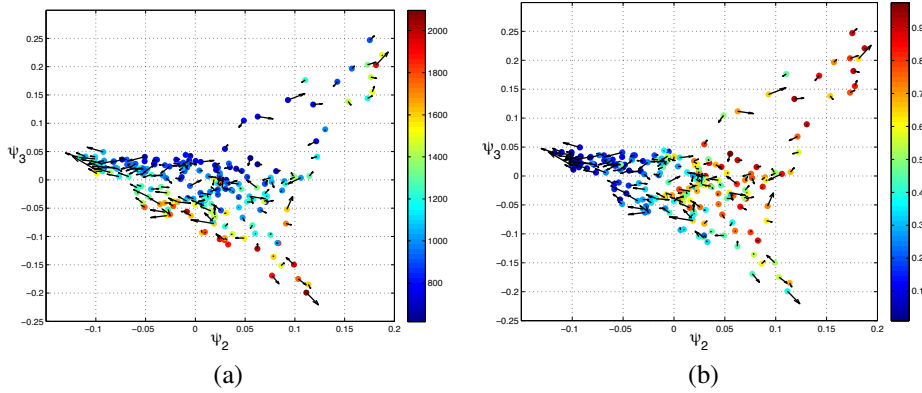

(a)                                                            (b)

Figure 3: Embedding the job market along with field $\mathbf{r} - 2\nabla U$ over the first two non-constant eigenvectors. The color map corresponds to the mean monthly wage in dollars (a) and to the female proportion (b) for each job.

We obtain an embedding of the job market that describes the relative position of jobs, their distribution, and the natural time progression from each job. Of these, the relative position and natural time progression are the most interesting. Together, they summarize the job market dynamics by describing which jobs are naturally "close" as well as where they can lead in the future. From a public policy perspective, this can potentially improve focus on certain jobs for helping individuals attain better upward mobility.

The job market was found to be a high dimensional manifold. We present only the first two dimensions, that is, the second and third eigenvectors of $H_{ss}^{(0)}$, since the first eigenvector is uninformative (constant) by construction. The eigenvectors showed correlation with important demographic data, such as wages and gender. Figure 3 displays this two-dimensional sub-embedding along with the directional information $\mathbf{r} - 2\nabla U$ for each dimension. The plot shows very little net progression toward regions of increasing mean salary[3]. This is somewhat surprising, but it is easy to overstate this observation: diffusion alone would be enough to move the individuals towards higher salary. What Figure 3 (a) suggests is that there appear to be no "external forces" advecting individuals towards higher salary. Nevertheless, there appear to be other external forces at play in the job market: Figure 3 (b), which is analogous to Figure 3 (a), but with gender replacing the salary color scheme, suggests that these forces push individuals towards greater gender differentiation. This is especially true amongst male-dominated jobs which appear to be advected toward the left edge of the embedding. Hence, this simple analysis of the job market can be seen as an indication that males and females tend to move away from each other over time, while neither seems to have a monopoly on high- or low- paying jobs.

## 6   Discussion

This paper makes three contributions: (1) it introduces a manifold-based generative model for directed graphs with weighted edges, (2) it obtains asymptotic results for operators constructed from the directed graphs, and (3) these asymptotic results lead to a natural algorithm for estimating the model.

**Generative Models** that assume that data are sampled from a manifold are standard for undirected graphs, but to our knowledge, none have yet been proposed for directed graphs. When $W$ is symmetric, it is natural to assume that it depends on the points' proximity. For asymmetric affinities $W$, one must include an additional component to explain the asymmetry. In the asymptotic limit, this is tantamount to defining a vector field on the manifold.

**Algorithm** We have used from [9] the idea of defining anisotropic kernels (indexed by $\alpha$) in order to separate the density $p$ and the manifold geometry $\mathcal{M}$. Also, we adopted their general assumptions about the symmetric part of the kernel. As a consequence, the recovery algorithm for $p$ and $\mathcal{M}$ is identical to theirs.

However, insofar as the *asymmetric part* of the kernel is concerned, everything, starting from the definition and the introduction of the vector field $\mathbf{r}$ as a way to model the asymmetry, through the derivation of the asymptotic expression for the symmetric plus asymmetric kernel, is new. We go significantly beyond the elegant idea of [9] regarding the use of anisotropic kernels by analyzing the four distinct renormalizations possible for a given $\alpha$, each of them combining different aspects of $\mathcal{M}, p$ and $\mathbf{r}$. Only the successful (and novel) combination of two different anisotropic operators is able to recover the directional flow $\mathbf{r}$.

Algorithm 1 is natural, but we do not claim it is the only possible one in the context of our model. For instance, we can also use $H_{sa}^{(\alpha)}$ to recover the operator $\nabla \cdot \mathbf{r}$ (which empirically seems to have worse numerical properties than $\mathbf{r} \cdot \nabla$). In the National Longitudinal Survery of Youth study, we were interested in the whole advective term, so we estimated it from a different combination of operators. Depending on the specific question, other features of the model could be obtained

**Limit Results** Proposition 3.1 is a general result on the asymptotics of asymmetric kernels. Recovering the manifold and $\mathbf{r}$ is just one, albeit the most useful, of the many ways of exploiting these results. For instance, $H_{sa}^{(0)}$ is the limit operator of the operators used in [3] and [5]. The limit analysis could be extended to other digraph embedding algorithms such as [4, 6].

**How general is our model?** Any kernel can be decomposed into a symmetric and an asymmetric part, as we have done. The assumptions on the symmetric part $h$ are standard. The paper of [7] goes one step further from these assumptions; we will discuss it in relationship with our work shortly. The more interesting question is how limiting are our assumptions regarding the choice of kernel, especially the asymmetric part, which we parameterized as $a_\epsilon(x,y) = \mathbf{r}/2 \cdot (y-x)h_\epsilon(x,y)$ in (4). In the asymptotic limit, this choice turns out to be fully general, at least up to the identifiable aspects of the model. For a more detailed discussion of this issue, see [8].

In [7], Ting, Huang and Jordan presented asymptotic results for a general family of kernels that includes asymmetric and random kernels. Our $k_\epsilon$ can be expressed in the notation of [7] by taking $w_x(y) \leftarrow 1 + \mathbf{r}(x,y) \cdot (y-x), r_x(y) \leftarrow 1, K_0 \leftarrow h, h \leftarrow \epsilon$. Their assumptions are more general than the assumptions we make here, yet our model is general up to what can be identified from $G$ alone. The distinction arises because [7] focuses on the graph construction methods from an observed sample of $\mathcal{M}$, while we focus on explaining an *observed directed graph* $G$ through a manifold generative process. Moreover, while the [7] results *can* be used to analyze data from directed graphs, they differ from our Proposition 3.1. Specifically, with respect to the limit in Theorem 3 from [7], we obtain the additional source terms $f(x)\nabla \cdot \mathbf{r}$ and $c(x)f(x)$ that follow from not enforcing conservation of mass while defining operators $H_{sa}^{(\alpha)}$ and $H_{as}^{(\alpha)}$.

We applied our theory of directed graph embedding to the analysis of the career sequences in Section 5, but asymmetric affinity data abound in other social contexts, and in the physical and life sciences. Indeed, any "similarity" score that is obtained from a likelihood of the form $W_{vu} =$likelihood$(u|v)$ is generally asymmetric. Hence our methods can be applied to study not only social networks, but also patterns of human movement, road traffic, and trade relations, as well as alignment scores in molecular biology. Finally, the physical interpretation of our model also makes it naturally applicable to physical models of flows.

### Acknowledgments

This research was partially supported by NSW awards IIS-0313339 and IIS-0535100.

## Footnotes

[1]The reader may notice that there are in fact eight possible combinations of kernel and degree distribution, since the anisotripic kernel (7) could also be defined using a symmetric or asymmetric outdegree distribution. However, there are only four distinct asymptotic results and they are all covered by using one kernel (symmetric or asymmetric) and one degree distribution (symmetric or asymmetric) throughout.

[2]A diffeomorphic embedding is guaranteed by using the eigendecomposition of $H_{ss}^{(1)}$.

[3]It is worth noting that in the NLSY data set, high paying jobs are teacher, nurse and mechanic. This is due to the fact that the career paths observed stop at at age 36, which is relatively early in an individual's career.

# References

[1] Belkin and Niyogi. Laplacian eigenmaps for dimensionality reduction and data representation. *Neural Computation*, 15:1373–1396, 2002.

[2] Nadler, Lafon, and Coifman. Diffusion maps, spectral clustering and eigenfunctions of fokker-planck operators. In *Neural Information Processing Systems Conference*, 2006.

[3] Meila and Pentney. Clustering by weighted cuts in directed graphs. In *SIAM Data Mining Conference*, 2007.

[4] Zhou, Huang, and Scholkopf. Learning from labeled and unlabeled data on a directed graph. In *International Conference on Machine Learning*, pages 1041–1048, 2005.

[5] Zhou, Schlkopf, and Hofmann. Semi-supervised learning on directed graphs. In *Advances in Neural Information Processing Systems*, volume 17, pages 1633–1640, 2005.

[6] Fan R. K. Chung. The diameter and laplacian eigenvalues of directed graphs. *Electr. J. Comb.*, 13, 2006.

[7] Ting, Huang, and Jordan. An analysis of the convergence of graph Laplacians. In *International Conference on Machine Learning*, 2010.

[8] Dominique Perrault-Joncas and Marina Meilă. Directed graph embedding: an algorithm based on continuous limits of laplacian-type operators. Technical Report TR 587, University of Washington - Department of Statistics, November 2011.

[9] Coifman and Lafon. Diffusion maps. *Applied and Computational Harmonic Analysis*, 21:6–30, 2006.

[10] Mikhail Belkin and Partha Niyogi. Convergence of laplacian eigenmaps. *preprint, short version NIPS 2008*, 2008.

[11] Coifman, Lafon, Lee, Maggioni, Warner, and Zucker. Geometric diffusions as a tool for harmonic analysis and structure definition of data: Diffusion maps. In *Proceedings of the National Academy of Sciences*, pages 7426–7431, 2005.

[12] United States Department of Labor. National longitudinal survey of youth 1979 cohort. http://www.bls.gov/nls/, retrived October 2011.

[13] Marc A. Scott. Affinity models for career sequences. *Journal of the Royal Statistical Society: Series C (Applied Statistics)*, 60(3):417–436, 2011.

